# Network Structuring And Training Using Rule-based Knowledge

**Volker Tresp**
Siemens AG
Central Research
Otto-Hahn-Ring 6
8000 München 83, Germany

**Jürgen Hollatz**[*]
Institut für Informatik
TU München
Arcisstraße 21
8000 München 2, Germany

**Subutai Ahmad**
Siemens AG
Central Research
Otto-Hahn-Ring 6
8000 München 83, Germany

## Abstract

We demonstrate in this paper how certain forms of rule-based knowledge can be used to prestructure a neural network of normalized basis functions and give a probabilistic interpretation of the network architecture. We describe several ways to assure that rule-based knowledge is preserved during training and present a method for complexity reduction that tries to minimize the number of rules and the number of conjuncts. After training the refined rules are extracted and analyzed.

## 1 INTRODUCTION

Training a network to model a high dimensional input/output mapping with only a small amount of training data is only possible if the underlying map is of low complexity and the network, therefore, can be of low complexity as well. With increasing

---

[*]Mail address: Siemens AG, Central Research, Otto-Hahn-Ring 6, 8000 München 83.

network complexity, parameter variance increases and the network prediction becomes less reliable. This predicament can be solved if we manage to incorporate prior knowledge to bias the network as it was done by Röscheisen, Hofmann and Tresp (1992). There, prior knowledge was available in the form of an algorithm which summarized the engineering knowledge accumulated over many years. Here, we consider the case that prior knowledge is available in the form of a set of rules which specify knowledge about the input/output mapping that the network has to learn. This is a very common occurrence in industrial and medical applications where rules can be either given by experts or where rules can be extracted from the existing solution to the problem.

The inclusion of prior knowledge has the additional advantage that if the network is required to extrapolate into regions of the input space where it has not seen any training data, it can rely on this prior knowledge. Furthermore, in many on-line control applications, the network is required to make reasonable predictions right from the beginning. Before it has seen sufficient training data it has to rely primarily on prior knowledge.

This situation is also typical for human learning. If we learn a new skill such as driving a car or riding a bicycle, it would be disastrous to start without prior knowledge about the problem. Typically, we are told some basic rules, which we try to follow in the beginning, but which are then refined and altered through experience. The better our initial knowledge about a problem, the faster we can achieve good performance and the less training is required (Towel, Shavlik and Noordewier, 1990).

## 2    FROM KNOWLEDGE TO NETWORKS

We consider a neural network $y = \mathcal{NN}(\mathbf{x})$ which makes a prediction about the state of $y \in \Re$ given the state of its input $\mathbf{x} \in \Re^n$. We assume that an expert provides information about the same mapping in terms of a set of rules. The premise of a rule specifies the conditions on $\mathbf{x}$ under which the conclusion can be applied. This region of the input space is formally described by a basis function $b_i(\mathbf{x})$. Instead of allowing only binary values for a basis function (1: premise is valid, 0: premise is not valid), we permit continuous positive values which represent the certainty or weight of a rule given the input.

We assume that the conclusion of the rule can be described in form of a mathematical expression, such as *conclusion$_i$: the output is equal to $w_i(\mathbf{x})$* where $w_i(\mathbf{x})$ is a function of the input (or a subset of the input) and can be a constant, a polynomial or even another neural network.

Since several rules can be active for a given state of the input, we define the output of the network to be a weighted average of the conclusions of the active rules where the weighting factor is proportional to the activity of the basis function given the input

$$y(\mathbf{x}) = \mathcal{NN}(\mathbf{x}) = \frac{\sum_i w_i(\mathbf{x})\, b_i(\mathbf{x})}{\sum_j b_j(\mathbf{x})}. \tag{1}$$

This is a very general concept since we still have complete freedom to specify the form of the basis function $b_i(\mathbf{x})$ and the conclusion $w_i(\mathbf{x})$. If $b_i(\mathbf{x})$ and $w_i(\mathbf{x})$ are

described by neural networks themselves, there is a close relationship with the adaptive mixtures of local experts (Jacobs, Jordan, Nowlan and Hinton, 1991). On the other hand, if we assume that the basis function can be approximated by a multivariate Gaussian

$$b_i(\mathbf{x}) = \kappa_i \ exp[-\frac{1}{2} \sum_j \frac{(x_j - \mu_{ij})^2}{\sigma_{ij}^2}], \qquad (2)$$

and if the $w_i$ are constants, we obtain the network of normalized basis functions which were previously described by Moody and Darken (1989) and Specht (1990).

In some cases the expert might want to formulate the premise as simple logical expressions. As an example, the rule

$$IF \ [((x_1 \approx a) \ AND \ (x_4 \approx b)] \ OR \ (x_2 \approx c) \ THEN \ y = d \times x^2$$

is encoded as

$$premise_i : \quad b_i(\mathbf{x}) = exp[-\frac{1}{2}\frac{(x_1 - a)^2 + (x_4 - b)^2}{\sigma^2}] + exp[-\frac{1}{2}\frac{(x_2 - c)^2}{\sigma^2}]$$

$$conclusion_i : \quad w_i(\mathbf{x}) = d \times x^2.$$

This formulation is related to the fuzzy logic approach of Tagaki and Sugeno (1992).

## 3    PRESERVING THE RULE-BASED KNOWLEDGE

Equation 1 can be implemented as a network of normalized basis functions $\mathcal{NN}^{init}$ which describes the rule-based knowledge and which can be used for prediction. Actual training data can be used to improve network performance. We consider four different ways to ensure that the expert knowledge is preserved during training.

*Forget.* We use the data to adapt $\mathcal{NN}^{init}$ with gradient descent (we typically adapt all parameters in the network). The sooner we stop training, the more of the initial expert knowledge is preserved.

*Freeze.* We freeze the parameters in the initial network and introduce a new basis function whenever prediction and data show a large deviation. In this way the network learns an additive correction to the initial network.

*Correct.* Whereas normal weight decay penalizes the deviation of a parameter from zero, we penalize a parameter if it deviates from its initial value $q_j^{init}$

$$E_P = \frac{1}{2}\alpha_j \sum_j (q_j - q_j^{init})^2 \qquad (3)$$

where the $q_j$ is a generic network parameter.

*Internal teacher.*   We formulate a penalty in terms of the mapping rather than in terms of the parameters

$$E_P = \frac{1}{2}\alpha \int (\mathcal{NN}^{init}(\mathbf{x}) - \mathcal{NN}(\mathbf{x}))^2 d\mathbf{x}.$$

This has the advantage that we do not have to specify priors on relatively unintuitive network parameters. Instead, the prior directly reflects the certainty that we

associate with the mapping of the initialized network which can often be estimated. Röscheisen, Hofmann and Tresp (1992) estimated this certainty from problem specific knowledge. We can approximate the integral in Equation 3 numerically by Monte-Carlo integration which leads to a training procedure where we adapt the network with a mixture of measured training data and training data artificially generated by $\mathcal{N}\mathcal{N}^{init}(\mathbf{x})$ at randomly chosen inputs. The mixing proportion directly relates to the weight of the penalty, $\alpha$ (Röscheisen, Hofmann and Tresp, 1992).

## 4   COMPLEXITY REDUCTION

After training the rules can be extracted again from the network but we have to ensure that the set of rules is as concise as possible, otherwise the value of the extracted rules is limited. We would like to find the smallest number of rules that can still describe the knowledge sufficiently. Also, the network should be encouraged to find rules with the smallest number of conjuncts, which in this case means that a basis function is only dependent on a small number of input dimensions.

We suggest the following pruning strategy for Gaussian basis functions.

1. *Prune basis functions.* Evaluate the relative weight of each basis function at its center $\omega_i = b_i(\mu_i)/\sum_j b_j(\mu_i)$ which is a measure of its importance in the network. Remove the unit with the smallest $\omega_i$. Figure 1 illustrates the pruning of basis functions.

2. *Prune conjuncts.* Successively, set the largest $\sigma_{ij}$ equal to infinity, effectively removing input $j$ from basis function $i$.

Sequentially remove basis functions and conjuncts until the error increases above a threshold. Retrain after a unit or a conjunct is removed.

## 5   A PROBABILISTIC INTERPRETATION

One of the advantages of our approach is that there is a probabilistic interpretation of the system. In addition, if the expert formulates his or her knowledge in terms of probability distributions then a number of useful properties can be derived (it is natural here to interpret probability as a subjective degree of belief in an event.). We assume that the system can be in a number of states $s_i$ which are unobservable. Formally, each of those hidden states corresponds to a rule. The prior probability that the system is in state $s_i$ is equal to $P(s_i)$. Assuming that the system is in state $s_i$ there is a probability distribution $P(\mathbf{x}, y|s_i)$ that we measure an input vector $\mathbf{x}$ and an output $y$ and

$$P(\mathbf{x}, y) = \sum_i P(\mathbf{x}, y|s_i)\, P(s_i) = \sum_i P(y|\mathbf{x}, s_i)\, P(\mathbf{x}|s_i)\, P(s_i). \qquad (4)$$

For every rule the expert specifies the probability distributions in the last sum. Let's consider the case that $P(\mathbf{x}, y|s_i) = P(\mathbf{x}|s_i)\, P(y|s_i)$ and that $P(\mathbf{x}|s_i)$ and $P(y|s_i)$ can be approximated by Gaussians. In this case Equation 4 describes a Gaussian mixture model. For every rule, the expert has to specify

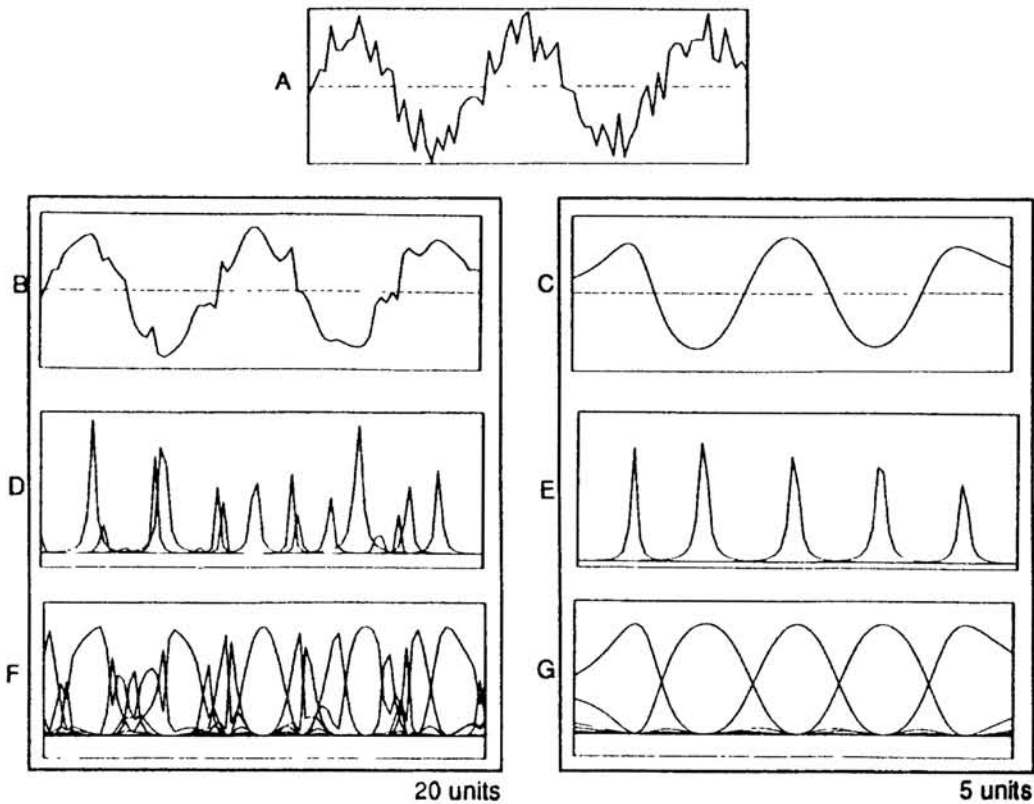

Figure 1: 80 values of a noisy sinusoid (A) are presented as training data to a network of 20 (Cauchy) basis functions, $(b_i(\mathbf{x}) = \kappa_i \, [1 + \sum_j (x_j - \mu_{ij})^2 / \sigma_{ij}^2]^{-2})$. (B) shows how this network also tries to approximate the noise in the data. (D) shows the basis functions $b_i(\mathbf{x})$ and (F) the normalized basis functions $b_i(\mathbf{x}) / \sum_j b_j(\mathbf{x})$. Pruning reduces the network architecture to 5 units placed at the extrema of the sinusoid (basis functions: E, normalized basis functions: G). The network output is shown in (C).

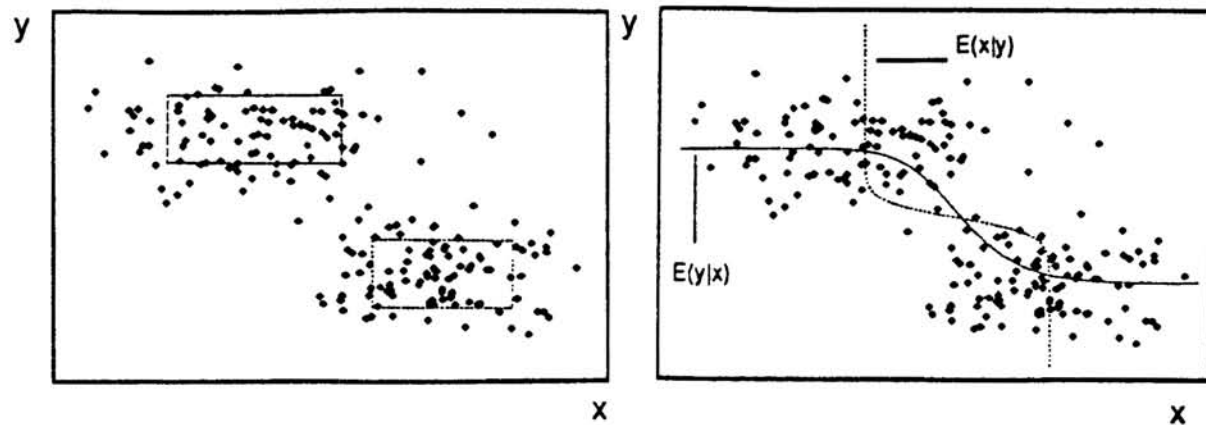

Figure 2: Left: The two rectangles indicate centers and standard deviations of two Gaussians that approximate a density. Right: the figure shows the expected values $\mathcal{E}(y|x)$ (continuous line) and $\mathcal{E}(x|y)$ (dotted line).

- $P(s_i)$, the probability of the occurrence of state $s_i$ (the overall weight of the rule),
- $P(\mathbf{x}|s_i) = N_i(\mathbf{x}; \mu_i, \Sigma_i)$, the probability that an input vector $\mathbf{x}$ occurs, given that the system is in state $s_i$, and
- $P(y|s_i) = N_i^y(y; w_i, \sigma_i^y)$, the probability of output $y$ given state $s_i$.

The evidence for a state given an input $\mathbf{x}$ becomes

$$P(s_i|\mathbf{x}) = \frac{P(\mathbf{x}|s_i)P(s_i)}{\sum_j P(\mathbf{x}|s_j)P(s_j)}$$

and the expected value of the output

$$\mathcal{E}(y|\mathbf{x}) = \frac{\sum_i \int y \, P(y|\mathbf{x}, s_i) \, dy \, P(\mathbf{x}|s_i)P(s_i)}{\sum_j P(\mathbf{x}|s_j)P(s_j)}, \tag{5}$$

where, $P(\mathbf{x}|s_i) = \int P(\mathbf{x}, y|s_i) \, dy$. If we substitute $b_i(\mathbf{x}) = P(\mathbf{x}|s_i)P(s_i)$ and $w_i(\mathbf{x}) = \int y \, P(y|\mathbf{x}, b_i) \, dy$ we can calculate the expected value of $y$ using the same architecture as described in Equation 1.

Subsequent training data can be employed to improve the model. The likelihood of the data $\{\mathbf{x}^k, y^k\}$ becomes

$$L = \prod_k \sum_i P(\mathbf{x}^k, y^k|s_i) \, P(s_i)$$

which can be maximized using gradient descent or EM. These adaptation rules are more complicated than supervised learning since according to our model the data generating process also makes assumptions about the distributions of the data in the input space.

Equation 4 gives an approximation of the joint probability density of input and output. Input and output are formally equivalent (Figure 2) and, in the case of Gaussian mixtures, we can easily calculate the optimal output given just a subset of inputs (Ahmad and Tresp, 1993).

A number of authors used clustering and Gaussian approximation on the input space alone and resources were distributed according to the complexity of the input space. In this method, resources are distributed according to the complexity of both input and output space.[1]

## 6   CLASSIFICATION

A conclusion now specifies the correct class. Let $\{b_{ik}|i = 1...N_k\}$ denote the set of basis functions whose conclusion specifies $class_k$. We set $w_{ij}^k = \delta_{kj}$, where $w_{ij}^k$ is the weight from basis function $b_{ij}$ to the $k$th output and $\delta_{kj}$ is the Kronecker symbol. The $k$th output of the network

$$y_k(\mathbf{x}) = \mathcal{NN}_k(\mathbf{x}) = \frac{\sum_{ij} w_{ij}^k b_{ij}(\mathbf{x})}{\sum_{lm} b_{lm}(\mathbf{x})} = \frac{\sum_i b_{ik}(\mathbf{x})}{\sum_{lm} b_{lm}(\mathbf{x})}. \tag{6}$$

specifies the certainty of $class_k$, given the input. During training, we do *not* adapt the output weights $w_{ij}^k$. Therefore, the outputs of the network are always positive and sum to one.

A probabilistic interpretation can be found if we assume that $P(\mathbf{x}|class_k)P(class_k)$ $\approx \sum_i b_{ik}(\mathbf{x})$. We obtain,

$$P(class_k|\mathbf{x}) = \frac{P(\mathbf{x}|class_k)P(class_k)}{\sum_l P(\mathbf{x}|class_l)P(class_l)}$$

and recover Equation 6. If the basis functions are Gausssians, again we obtain a Gaussian mixture learning problem and, as a special case (one unit per class), a Gaussian classifier.

# 7    APPLICATIONS

We have validated our approach on a number of applications including a network that learned how to control a bicycle and an application in the legal sciences (Hollatz and Tresp, 1992). Here we present results for a well known data set, the Boston housing data (Breiman *et al.*, 1981), and demonstrate pruning and rule extraction. The task is to predict the housing price in a Boston neighborhood as a function of 13 potentially relevant input features. We started with 20 Gaussian basis functions which were adapted using gradient descent. We achieved a generalization error of 0.074. We then pruned units and conjuncts according to the procedure described in Section 4. We achieved the best generalization error (0.058) using 4 units (this is approximately 10% better than the result reported for CART in Breiman *et al.*, 1981). With only two basis functions and 3 conjuncts, we still achieved reasonable prediction accuracy (generalization error of 0.12; simply predicting the mean results in a generalization error of 0.28). Table 1 describes the final network. Interestingly, our network was left with the input features which CART also considered the most relevant.

The network was trained with normalized inputs. If we translate them back into real world values, we obtain the rules:

*Rule*$_{14}$: IF the number of rooms (RM) is approximately 5.4 (0.62 corresponds to 5.4 rooms which is smaller than the average of 6.3) AND the pupil/teacher value is approximately 20.2 (0.85 corresponds to 20.2 pupils/teacher which is higher than the average of 18.4) THEN the value of the home is approximately \$14000 (0.528 corresponds to \$14000 which is lower than the average of \$22500).

Table 1: Network structure after pruning.

| | conclusion | feature $j$ | CART rating | center: $\mu_{ij}$ | width: $\sigma_{ij}$ |
|---|---|---|---|---|---|
| Unit#: $i = 14$ $\kappa_i = 0.17$ | $w_i = 0.528$ | RM P/T | second third | 0.62 0.85 | 0.21 0.35 |
| Unit#: $i = 20$ $\kappa_i = 0.83$ | $w_i = 1.6$ | LSTAT | most important | 0.06 | 0.24 |

$Rule_{20}$: IF the percentage of lower-status population (LSTAT) is approximately 2.5% (0.06 corresponds to 2.5% which is lower than the average of 12.65%), THEN the value of the home is approximately $34000 (1.6 corresponds to $34000 which is higher than the average of $22500).

## 8   CONCLUSION

We demonstrated how rule-based knowledge can be incorporated into the structuring and training of a neural network. Training with experimental data allows for rule refinement. Rule extraction provides a quantitative interpretation of what is "going on" in the network, although, in general, it is difficult to define the domain where a given rule "dominates" the network response and along which boundaries the rules partition the input space.

### Acknowledgements

We acknowledge valuable discussions with Ralph Neuneier and his support in the Boston housing data application. V.T. was supported in part by a grant from the Bundesminister für Forschung und Technologie and J. H. by a fellowship from Siemens AG.

## Footnotes

[1]Note, that a probabilistic interpretation is only possible if the integral over a basis function is finite, i.e. all variances are finite.

### References

S. Ahmad and V. Tresp. Some solutions to the missing feature problem in vision. This volume, 1993.

L. Breiman *et al.*. *Classification and regression trees.*   Wadsworth and Brooks, 1981.

R. A. Jacobs, M. I. Jordan, S. J. Nowlan and G. E. Hinton. Adaptive mixtures of local experts. *Neural Computation*, Vol. 3, pp. 79-87, 1991.

J. Hollatz and V. Tresp.   A Rule-based network architecture. *Artificial Neural Networks II*, I. Aleksander, J. Taylor, eds., Elsevier, Amsterdam, 1992.

J. Moody and C. Darken.   Fast learning in networks of locally-tuned processing units. *Neural Computation*, Vol. 1, pp. 281-294, 1989.

M. Röscheisen, R. Hofmann and V. Tresp. Neural control for rolling mills: incorporating domain theories to overcome data deficiency. In: *Advances in Neural Information Processing Systems 4*, 1992.

D. F. Specht. Probabilistic neural networks. *Neural Networks*, Vol. 3, pp. 109-117, 1990.

T. Takagi and M. Sugeno. Fuzzy identification of systems and its applications to modeling and control. *IEEE Transactions on Systems, Man and Cybernetics*, Vol. 15, No. 1, pp. 116-132, 1985.

G. G. Towell, J. W. Shavlik and M. O. Noordewier. Refinement of approximately correct domain theories by knowledge-based neural networks. In *Proceedings of the Eights National Conference on Artificial Intelligence*, pp. 861-866, MA, 1990.